# Approximating Equilibria in Sequential Auctions with Incomplete Information and Multi-Unit Demand

**Amy Greenwald and Eric Sodomka**
Department of Computer Science
Brown University
Providence, RI 02912
{amy,sodomka}@cs.brown.edu

**Jiacui Li**
Department of Applied Math/Economics
Brown University
Providence, RI 02912
jiacui_li@alumni.brown.edu

## Abstract

In many large economic markets, goods are sold through sequential auctions. Examples include eBay, online ad auctions, wireless spectrum auctions, and the Dutch flower auctions. In this paper, we combine methods from game theory and decision theory to search for approximate equilibria in sequential auction domains, in which bidders do not know their opponents' values for goods, bidders only partially observe the actions of their opponents', and bidders demand multiple goods. We restrict attention to two-phased strategies: first predict (i.e., learn); second, optimize. We use best-reply dynamics [4] for prediction (i.e., to predict other bidders' strategies), and then assuming fixed other-bidder strategies, we estimate and solve the ensuing Markov decision processes (MDP) [18] for optimization. We exploit auction properties to represent the MDP in a more compact state space, and we use Monte Carlo simulation to make estimating the MDP tractable. We show how equilibria found using our search procedure compare to known equilibria for simpler auction domains, and we approximate an equilibrium for a more complex auction domain where analytical solutions are unknown.

## 1 Introduction

Decision-making entities, whether they are businesses, governments, or individuals, usually interact in game-theoretic environments, in which the final outcome is intimately tied to the actions taken by others in the environment. Auctions are examples of such game-theoretic environments with significant economic relevance. Internet advertising, of which a significant portion of transactions take place through online auctions, has had spending increase 24 percent from 2010 to 2011, globally becoming an $85 billion industry [16]. The FCC has conducted auctions for wireless spectrum since 1994, reaching sales of over $60 billion.[1] Perishable commodities such as flowers are often sold via auction; the Dutch flower auctions had about $5.4 billion in sales in 2011.[2]

A game-theoretic equilibrium, in which each bidder best responds to the strategies of its opponents, can be used as a means of prescribing and predicting auction outcomes. Finding equilibria in auctions is potentially valuable to bidders, as they can use the resulting strategies as prescriptions that guide their decisions, and to auction designers, as they can use the resulting strategies as predictions for bidder behavior. While a rich literature exists on computing equilibria for relatively simple auction games [11], auction theory offers few analytical solutions for real-world auctions. Even existing computational methods for approximating equilibria quickly become intractable as the number of bidders and goods, and the complexity of preferences and decisions, increase.

In this paper, we combine methods from game theory and decision theory to approximate equilibria in sequential auction domains, in which bidders do not know their opponents' values for goods, bidders partially observe the actions of their opponents', and bidders demand multiple goods. Our method of searching for equilibria is motivated by the desire to reach strategies that real-world bidders might actually use. To this end, we consider strategies that consist of two parts: a prediction (i.e., learning) phase and an optimization phase. We use best-reply dynamics [4] for prediction (i.e., to predict other bidders' strategies), and then assuming fixed other-bidder strategies, we estimate and solve a Markov decision processes (MDP) [18] for optimization. We exploit auction properties to represent the MDPs in a more compact state space, and we use Monte Carlo simulation to make estimating the MDPs tractable.

## 2  Sequential Auctions

We focus on sequential sealed-bid auctions, with a single good being sold at each of $K$ rounds. The number of bidders $n$ and the order in which goods are sold are assumed to be common knowledge. During auction round $k$, each bidder $i$ submits a private bid $b_i^k \in B_i$ to the auctioneer. We let $b^k = \langle b_1^k, \ldots, b_n^k \rangle$ denote the vector of bids submitted by all bidders at round $k$. The bidder who submits the highest bid wins and is assigned a cost based on a commonly known payment rule.

At the end of round $k$, the auctioneer sends a private (or public) signal $o_i^k \in O_i$ to each bidder $i$, which is a tuple specifying information about the auction outcome for round $k$, such as the winning bid, the bids of all agents, the winner identities, whether or not a particular agent won the good, or any combination thereof. Bidders only observe opponents' bids if those bids are announced by the auctioneer. Regardless, we assume that bidder $i$ is told at least which set of goods she won in the $k$th round, $w_i^k \in \{\emptyset, \{k\}\}$, and how much she paid, $c_i^k \in \mathbb{R}$. We let $\psi(o^k \mid b^k) \in [0,1]$ denote the probability that the auctioneer sends the bidders signals $o^k = \langle o_1^k, \ldots, o_n^k \rangle$ given $b^k$, and we let $\psi(o_i^k \mid b^k)$ express the probability that player $i$ receives signal $o_i^k$, given $b^k$.

An auction history at round $k$ consists of past bids plus all information communicated by the auctioneer though round $k-1$. Let $h_i^k = \langle (b_i^1, o_i^1), \ldots, (b_i^{k-1} o_i^{k-1}) \rangle$ be a possible auction *history* at round $k$ as observed by bidder $i$. Let $H_i$ be the set of all possible auction histories for bidder $i$. Each bidder $i$ is endowed with a privately known *type* $\theta_i \in \Theta_i$, drawn from a commonly known distribution $F$, that determines bidder $i$'s *valuations* for various bundles of goods. A (behavioral) *strategy* $\sigma_i : \Theta \times H_i \mapsto \triangle B_i$ for bidder $i$ specifies a distribution over bids for each possible type and auction history. The set $\Sigma_i$ contains all possible strategies.

At the end of the $K$ auction rounds, bidder $i$'s *utility* is based on the bundle of goods she won and the amount she paid for those goods. Let $X \subseteq \{1, \ldots, K\}$ be a possible bundle of goods, and let $v(X; \theta_i)$ denote a bidder's valuation for bundle $X$ when its type is $\theta_i$. No assumptions are made about the structure of this value function. A bidder's utility for type $\theta_i$ and history $h^K$ after $K$ auction rounds is simply that bidder's value for the bundle of goods it won minus its cost: $u_i(\theta_i, h_i^K) = v(\cup_{k=1}^K w_i^k; \theta_i) - \sum_{k=1}^K c_i^k$.

Given a sequential auction $\Gamma$ (defined by all of the above), bidder $i$'s objective is to choose a strategy that maximizes its *expected utility*. But this quantity depends on the actions of other bidders. A *strategy profile* $\vec{\sigma} = (\sigma_1, \cdots, \sigma_N) = (\sigma_i, \sigma_{-i})$ defines a strategy for each bidder. (Throughout the paper, subscript $i$ refers to a bidder $i$ while $-i$ refers to all bidders except $i$.) Let $U_i(\vec{\sigma}) = \mathbb{E}_{\theta_i, h_i^K \mid \vec{\sigma}}[u_i(\theta_i, h_i^K)]$ denote bidder $i$'s expected utility given strategy profile $\vec{\sigma}$.

**Definition 1** ($\epsilon$-Bayes-Nash Equilibrium ($\epsilon$-BNE))**.** Given a sequential auction $\Gamma$, a strategy profile $\vec{\sigma} \in \Sigma$ is an *$\epsilon$-Bayes-Nash-equilibrium* if $U_i(\vec{\sigma}) + \epsilon \geq U_i(\sigma_i', \sigma_{-i}) \quad \forall i \in \{1, \ldots, n\}, \forall \sigma_i' \in \Sigma_i$.

In an *$\epsilon$-Bayes-Nash Equilibrium*, each bidder has to come within an additive factor ($\epsilon$) of best-responding to its opponent strategies. A *Bayes-Nash equilibrium* is an $\epsilon$-Bayes-Nash equilibrium where $\epsilon = 0$. In this paper, we explore techniques for finding $\epsilon$-BNE in sequential auctions. We also explain how to experimentally estimate the so-called $\epsilon$-factor of a strategy profile:

**Definition 2** ($\epsilon$-Factor)**.** Given a sequential auction $\Gamma$, the $\epsilon$-factor of strategy profile $\vec{\sigma}$ for bidder $i$ is $\epsilon_i(\vec{\sigma}) = \max_{\sigma_i'} U_i(\sigma_i', \sigma_{-i}) - U_i(\sigma_i, \sigma_{-i})$. In words, the $\epsilon$-factor measures bidder $i$'s loss in expected utility for not playing his part of $\vec{\sigma}$ when other bidders are playing their parts.

# 3 Theoretical Results

As the number of rounds, bidders, possible types, or possible actions in a sequential auction increases, it quickly becomes intractable to find equilibria using existing computational methods. Such real-world intractability is one reason bidders often do not attempt to solve for equilibria, but rather optimize with respect to predictions about opponent behavior. Building on past work [2, 8], our first contribution is to fully represent the decision problem for a single bidder $i$ in a sequential auction $\Gamma$ as a Markov decision process (MDP).

**Definition 3** (Full-history MDP). A *full-history MDP* $M_i(\Gamma, \theta_i, T)$ represents the sequential auction $\Gamma$ from bidder $i$'s perspective, assuming $i$'s type is $\theta_i$, with states $S = H_i$, actions $A = B_i$, rewards $R(s) = \{u_i(\theta_i, h_i^K)$ if $s = h_i^K$ is a history of length $K$; 0 otherwise$\}$, and transition function $T$.

If bidder types are correlated, bidder $i$'s type informs its beliefs about opponents' types and thus opponents' predicted behavior. For notational and computational simplicity, we assume that bidder types are drawn independently, in which case there is one transition function $T$ regardless of bidder $i$'s type. We also assume that bidders are *symmetric*, meaning their types are all drawn from the same distribution. When bidders are symmetric, we can restrict our attention to symmetric equilibria, where a single set of full-history MDPs, one per type, is solved on behalf of all bidders.

**Definition 4** (MDP Assessment). An *MDP assessment* $(\pi, T)$ for a sequential auction $\Gamma$ is a set of policies $\{\pi^{\theta_i} \mid \theta_i \in \Theta_i\}$, one for each full-history MDP $M_i(\Gamma, \theta_i, T)$.

We now explain where the transition function $T$ comes from. At a high level, we define (symmetric) *induced transition probabilities* $\texttt{Induced}(\pi)$ to be the transition probabilities that result from agent $i$ using Bayesian updating to infer something about its opponents' private information, and then reasoning about its opponents' subsequent actions, assuming they all follow policy $\pi$. The following example provides some intuition for this process.

**Example 1.** Consider a first-price sequential auction with two rounds, two bidders, two possible types ("H" and "L") drawn independently from a uniform prior (i.e., $p(H) = 0.5$ and $p(L) = 0.5$), and two possible actions ("high" and "low"). Suppose Bidder 2 is playing the following simple strategy: if type H: bid "high" with probability .9, and bid "low" with probability .1; if type L: bid "high" with probability .1, and bid "low" with probability .9.

At round $k = 1$, from the perspective of Bidder 1, the only uncertainty that exists is about Bidder 2's type. Bidder 1's beliefs about Bidder 2's type is based solely on the type prior, resulting in beliefs that Bidder 2 will bid "high" and "low" each with equal probability. Suppose Bidder 1 bids "low" and loses to Bidder 2, who the auctioneer reports as having bid "high". At round $k = 2$, Bidder 1 must update its posterior beliefs about Bidder 2 after observing the given outcome. This is done using Bayes' rule to find that Bidder 2 is of type "H" with probability 0.9. Based on its policy, in the subsequent round, the probability Bidder 2 bids "high" is $0.9(0.9) + 0.1(0.1) = 0.82$, and the probability it bids "low" is $0.9(0.1) + 0.1(0.9) = 0.18$. Given this bid distribution for Bidder 2, Bidder 1 can compute her probability of transitioning to various future states for each possible bid.

More formally, denoting $s_i^k$ and $a_i^k$ as agent $i$'s state and action at auction round $k$, respectively, define $\Pr(s_i^{k+1} \mid s_i^k, a_i^k)$ to be the probability of reaching state $s_i^{k+1}$ given that action $a_i^k$ was taken in state $s_i^k$. By twice applying the law of total probability and then noting conditional independencies,

$$
\begin{aligned}
\Pr(s_i^{k+1} \mid s_i^k, a_i^k) &= \sum_{a_{-i}^k} \Pr(s_i^{k+1} \mid s_i^k, a_i^k, a_{-i}^k) \Pr(a_{-i}^k \mid s_i^k, a_i^k) \\
&= \sum_{\theta_{-i}} \sum_{s_{-i}^k} \sum_{a_{-i}^k} \Pr(s_i^{k+1} \mid s_i^k, a_i^k, a_{-i}^k, s_{-i}^k, \theta_{-i}) \Pr(a_{-i}^k \mid s_i^k, a_i^k, s_{-i}^k, \theta_{-i}) \Pr(s_{-i}^k, \theta_{-i} \mid s_i^k, a_i^k) \\
&= \sum_{\theta_{-i}} \sum_{s_{-i}^k} \sum_{a_{-i}^k} \underbrace{\Pr(s_i^{k+1} \mid s_i^k, a_i^k, a_{-i}^k)} \underbrace{\Pr(a_{-i}^k \mid s_{-i}^k, \theta_{-i})} \underbrace{\Pr(s_{-i}^k, \theta_{-i} \mid s_i^k, a_i^k)}
\end{aligned}
\tag{1}
$$

The first term in Equation 1 is defined by the auction rules and depends only on the actions taken at round $k$: $\Pr(s_i^{k+1} \mid s_i^k, a_i^k, a_{-i}^k) = \psi(o_i^k \mid a^k)$. The second term is a joint distribution over opponents' actions given opponents' private information. Each agent's action at round $k$ is conditionally independent given that agent's state at round $k$: $\Pr(a_{-i}^k \mid s_{-i}^k, \theta_{-i}) = \prod_{j \neq i} \Pr(a_j^k \mid s_j^k, \theta_j) = \prod_{j \neq i} \pi^{\theta_j}(a_j^k \mid s_j^k)$. The third term is the joint distribution over opponents' private information,

given agent $i$'s observations. This term can be computed using Bayesian updating. We compute induced transition probabilities $\texttt{Induced}(\pi)(s_i^k, a_i^k, s_i^{k+1})$ using Equation 1.

**Definition 5** ($\delta$-Stable MDP Assessment)**.** An MDP assessment $(\pi, T)$ for a sequential auction $\Gamma$ is called $\delta$-*stable* if $d(T, \texttt{Induced}(\pi)) < \delta$, for some symmetric distance function $d$.

When $\delta = 0$, the induced transition probabilities exactly equal the transition probabilities from the MDP assessment $(\pi, T)$, meaning that if all agents follow $(\pi, T)$, the transition function $T$ is correct.

Define $U_i(\pi, T) \equiv \mathbb{E}_{\theta_i, h_i^K | \pi, T}[u_i(\theta_i, h_i^K)]$ to be the expected utility for following an MDP assessment's policy $\pi$ when the transition function is $T$. (We abbreviate $U_i$ by $U$ because of symmetry.)

**Definition 6** ($\alpha$-Optimal MDP Assessment)**.** An MDP assessment $(\pi, T)$ for a sequential auction $\Gamma$ is called $\alpha$-*optimal* if for all policies $\pi'$, $U(\pi, T) + \alpha \geq U(\pi', T)$.

If each agent is playing a 0-optimal (i.e., optimal) 0-stable (i.e., stable) MDP assessment for the sequential auction $\Gamma$, each agent is best responding to its beliefs, and each agent's beliefs are correct. It follows that *any optimal stable MDP assessment for the sequential auction $\Gamma$ corresponds to a symmetric Bayes-Nash equilibrium for $\Gamma$.* Corollary 2 (below) generalizes this observation to approximate equilibria.[3]

Suppose we have a black box that tells us the difference in perceived versus actual expected utility for optimizing with respect to the wrong beliefs: i.e., the wrong transition function. More precisely, if we were to give the black box two transition functions $T$ and $T'$ that differ by at most $\delta$ (i.e., $d(T, T') < \delta$), the black box would return $\max_\pi |U(\pi, T) - U(\pi, T')| \equiv D(\delta)$.

**Theorem 1.** *Given such a black box, if $(\pi, T)$ is an $\alpha$-optimal $\delta$-stable MDP assessment for the sequential auction $\Gamma$, then $\pi$ is a symmetric $\epsilon$-Bayes-Nash equilibrium for $\Gamma$, where $\epsilon = 2D(\delta) + \alpha$.*

*Proof.* Let $T_\pi = \texttt{Induced}(\pi)$, and let $\pi^*$ be such that $(\pi^*, T_\pi)$ is an optimal MDP assessment.

$$U(\pi, T_\pi) \geq U(\pi, T) - D(\delta) \tag{2}$$
$$\geq U(\pi^*, T) - (\alpha + D(\delta)) \tag{3}$$
$$\geq U(\pi^*, T_\pi) - (\alpha + 2D(\delta)) \tag{4}$$

Lines 2 and 4 hold because $(\pi, T)$ is $\delta$-stable. Line 3 holds because $(\pi, T)$ is $\alpha$-optimal.

**Corollary 2.** *If $(\pi, T)$ is an $\alpha$-optimal $\delta$-stable MDP assessment for the sequential auction $\Gamma$, then $\pi$ is a symmetric $\epsilon$-Bayes-Nash equilibrium for $\Gamma$, where $\epsilon = 2\delta K + \alpha$.*

In particlar, when the distance between other-agent bid predictions and the actual other-agent bids induced by the actual other-agent policies is less than $\delta$, optimizing agents play a $2\delta K$-BNE.

This corollary follows from the simulation lemma in Kakade et al. [9], which provides us with a black box.[4] In particular, if MDP assessment $(\pi, T)$ is $\delta$-stable, then $|U(\pi, T) - U(\pi, \texttt{Induced}(\pi))| \leq \delta K$, where $d(T, T') = \sum_{s_i^{k+1}} |T(s_i^k, a_i^k, s_i^{k+1}) - T'(s_i^k, a_i^k, s_i^{k+1})|$ and $K$ is the MDP's horizon.

Wellman et al. [24] show that, for simultaneous one-shot auctions, optimizing with respect to predictions about other-agent bids is an $\epsilon$-Bayes-Nash equilibrium, where $\epsilon$ depends on the distance between other-agent bid predictions and the actual other-agent bids induced by the actual other-agent strategies. Corollary 2 is an extension of that result to sequential auctions.

## 4   Searching for an $\epsilon$-BNE

We now know that an optimal, stable MDP assessment is a BNE, and moreover, a near-optimal, near-stable MDP assessment is nearly a BNE. Hence, we propose to search for approximate BNE by searching the space of MDP assessments for any that are nearly optimal and nearly stable.

Our search uses an iterative two-step learning process. We first find a set of optimal policies $\pi$ with respect to some transition function $T$ (i.e., $\pi = \texttt{Solve\_MDP}(T)$) using dynamic programming, as described by Bellman's equations [1]. We then update the transition function $T$ to reflect what would happen if all agents followed the new policies $\pi$ (i.e., $T^* = \texttt{Induced}(\pi)$). More precisely,

1. Initiate the search from an arbitrary MDP assessment $(\pi^0, T^0)$

2. Initialize $t = 1$ and $\epsilon = \infty$

3. While $(t < \tau)$ and $(\epsilon > \kappa)$

    (a) PREDICT: $T^t = \texttt{Induced}(\pi^{t-1})$
    (b) OPTIMIZE: for all types $\theta_i$, $\pi^t = \texttt{Solve\_MDP}(\theta_i, T^t)$
    (c) Calculate $\epsilon \equiv \epsilon(\pi^\tau)$ (defined below)
    (d) Increment $t$

4. Return MDP assessment $(\pi^\tau, T^\tau)$ and $\epsilon$

This learning process is not guaranteed to converge, so upon termination, it could return an optimal, $\delta$-stable MDP assessment for some very large $\delta$. However, it has been shown to be successful experimentally in simultaneous auction games [24] and other large games of imperfect information [7].

**Monte Carlo Simulations**   Recall how we define induced transition functions (Equation 1). In practice, the Bayesian updating involved in this calculation is intractable. Instead, we employ Monte Carlo simulations. First, we further simplify Equation 1 using the law of total probability and noting conditional independencies (Equation 5). Second, we exploit some special structure of sequential auctions: if nothing but the winning price at each round is revealed, conditional on reaching state $s_i^k$, the posterior distribution over highest opponent bids is sufficient for computing the probability of that round's outcome (Equation 6).[5]  Third, we simulate $N$ auction trajectories for the given policy $\pi$ and multiple draws from the agent's type distribution, and count the number of times each highest opponent bid occurs at each state (Equation 7):

$$\texttt{Induced}(\pi)(s_i^k, a_i^k, s_i^{k+1}) \;=\; \Pr(s_i^{k+1} \mid s_i^k, a_i^k, \max a_{-i}^k)\Pr(\max a_{-i}^k \mid s_i^k, a_i^k) \qquad (5)$$

$$\;=\; \Pr(s_i^{k+1} \mid s_i^k, a_i^k, \max a_{-i}^k)\Pr(\max a_{-i}^k \mid s_i^k) \qquad (6)$$

$$\texttt{Induced}_N(\pi)(s_i^k, a_i^k, s_i^{k+1}) \;=\; \psi(o_i^k \mid \max(a_{-i}^k), a_i^k)\frac{\#(\max(a_{-i}^k), s_i^k)}{\#(s_i^k)} \qquad (7)$$

**Solving the MDP**   As previously stated, we solve the MDPs exactly using dynamic programming, but we can only do so because we exploit the structure of auctions to reduce the number of states in each MDP. Recall that we assume symmetry: i.e., all bidders' types are drawn from the same distribution. Under this assumption, when the auctioneer announces that an Bidder $j$ has won an auction for the first time, this provides the same information as if a different Bidder $k$ won an auction for the first time. We thus collapse these two outcomes into the same state. This can greatly decrease the MDP state space, particularly if the number of players $n$ is larger than the number of auctions $K$, as is often the case in competitive markets. In fact, by handling this symmetry, the MDP state space is the same for any number of players $n \geq K$.[6]  Second, we exploit the property of *losing bid symmetry*: if a bidder $i$ loses with a bid of $b$ or a bid of $b'$, its beliefs about its opponents bids are unchanged, and thus it receives the same reward for placing the same bid at either resulting state.

_______________________

[5]A distribution over the next round's highest opponent bid is only sufficient without the possibility of ties. In ties can occur, a distribution over the number of opponents placing that highest bid is also needed. In our experiments, we do not maintain such a distribution; if there is a tie, the agent in question wins with probability 0.5 (i.e., we assume it tied with only one opponent).

[6]Even when $n < K$, the state space can still be significantly reduced, since instead of $n$ different possible winner identities in the $k$th round, there are only $\min(n; k + 1)$. In the extreme case of $n = 2$, there is no winner identity symmetry to exploit, since $n = k + 1$ even in the first round.

$\epsilon$**-factor Approximation**    Define $U_i(\vec{\pi}) = \mathbb{E}_{\theta_i, h_i^K | \vec{\pi}}[u_i(\theta_i, h_i^K)]$ to be bidder $i$'s expected utility $i$ when each agent plays its part in the vector of MDP assessment policies $\vec{\pi}$. Following Definition 2, the $\epsilon$-factor measures bidder $i$'s loss in expected utility for not playing his part of $\vec{\pi}$ when other bidders are playing their parts: $\epsilon_i(\vec{\pi}) = \max_{\pi'_i} U_i(\pi'_i, \pi_{-i}) - U_i(\pi_i, \pi_{-i})$. In fact, since we are only interested in finding symmetric equilibria, where $\vec{\pi} = (\pi, \dots, \pi)$, we calculate $\epsilon(\pi) = \max_{\pi'} U(\pi', \vec{\pi}_{-i}) - U(\pi, \vec{\pi}_{-i})$.

The first term in this definition is the expected utility of the best response, $\pi^*$, to $\vec{\pi}_{-i}$. This quantity typically cannot be computed exactly, so instead, we compute a near-best response $\hat{\pi}_N^* = \texttt{Solve\_MDP}(\texttt{Induced}_N(\pi))$, which is optimal with respect to $\texttt{Induced}_N(\pi) \approx \texttt{Induced}(\pi)$, and then measure the gain in expected utility of deviating from $\pi$ to $\hat{\pi}_N^*$.

Further, we approximate expected utility through Monte Carlo simulation. Specifically, we compute $\hat{U}_L(\vec{\pi}) = \frac{1}{L} \sum_{l=1}^L u(\theta^l, h^l)$ by sampling $\vec{\theta}$ and simulating $(\pi^\theta, \dots, \pi^\theta)$ $L$ times, and then averaging bidder $i$'s resulting utilities. Thus, we approximate $\epsilon(\pi)$ by $\hat{\epsilon}(\pi) \approx \hat{U}_L(\hat{\pi}_N^*, \vec{\pi}_{-i}) - \hat{U}_L(\pi, \vec{\pi}_{-i})$.

The approximation error in $\hat{\epsilon}(\pi)$ comes from both imprecision in $\texttt{Induced}_N(\pi)$, which depends on the sample size $N$, and imprecision in the expected utility calculation, which depends on the sample size $L$. The latter is $\mathcal{O}(\sqrt{L})$ by the central limit theorem, and can be made arbitrarily small. (In our experiments, we plot the confidence bounds of this error to make sure it is indeed small.) The former arises because $\hat{\pi}_N^*$ is not truly optimal with respect to $\texttt{Induced}(\pi)$, and goes to zero as $N$ goes to infinity by standard reinforcement learning results [20]. In practice we make sure that $N$ is large enough so that this error is negligible.

## 5    Experimental Results

This section presents the results of running our iterative learning method on three auction models studied in the economics literature: Katzman [10], Weber [23], and Menezes and Monteiro [14]. These models are all two-round, second-price, sequential auctions[7], with continuous valuation spaces; they differ only in their specific choice of valuations. The authors analytically derive a symmetric pure strategy equilibrium for each model, which we attempt to re-discover using our iterative method. After discretizing the valuation space, our method is sufficiently general to apply immediately in all three settings.

Although these particular sequential auctions are all second price, our method applies to sequential auctions with other rules as well. We picked this format because of the abundance of corresponding theoretical results and the simplicity of exposition in two-round auctions. It is a dominant strategy to bid truthfully in a one-shot second-price auction [22]; hence, when comparing policies in two-round second-price auctions it suffices to compare first-round policies only.

**Static Experiments**    We first run one iteration of our learning procedure to check whether the derived equilibria are strict. In other words, we check whether $\texttt{Solve\_MDP}(\texttt{Induced}_N(\pi^E)) = \pi^E$, where $\pi^E$ is a (discretized) derived equilibrium strategy. For each of the three models, Figures 1(a)–1(c) compare first-round bidding functions of the former (blue) with the latter (green).

Our results indicate that the equilibria derived by Weber and Katzman are indeed strict, while that by Menezes and Monteiro (MM) is not, since there exists a set of best-responses to the equilibrium strategy, not a unique best-response. We confirm analytically that the set of bids output by our learning procedure are best-responses to the theoretical equilibrium, with the upper bound being the known theoretical equilibrium strategy and the lower bound being the black dotted line.[8] To our knowledge, this instability was previously unknown.

**Dynamic Experiments**    Since MM's theoretical equilibrium is not strict, we apply our iterative learning procedure to search for more stable approximate equilibria. Our procedure converges within a small number of iterations to an $\epsilon$-BNE with a small $\epsilon$ factor, and the convergence is robust across different initializations. We chose initial strategies $\pi^0$ parametrized by $p \in \mathbb{R}^+$ that bid $x^p$ when the marginal value of winning an additional good is $x$. By varying the exponent $p$, we initialize the learning procedure with bidding strategies whose level of aggressiveness varies.

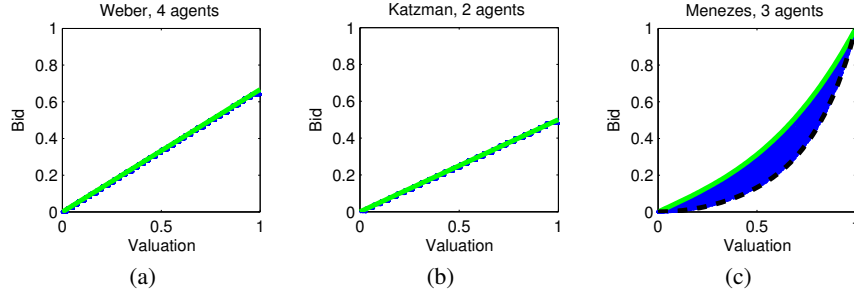

Figure 1: Comparison of first-round bidding functions of theoretical equilibrium strategies (green) and that of the best response from one step of the iterative learning procedure initialized with those equilibrium strategies (blue). (a) Weber. (b) Katzman. (c) MM.

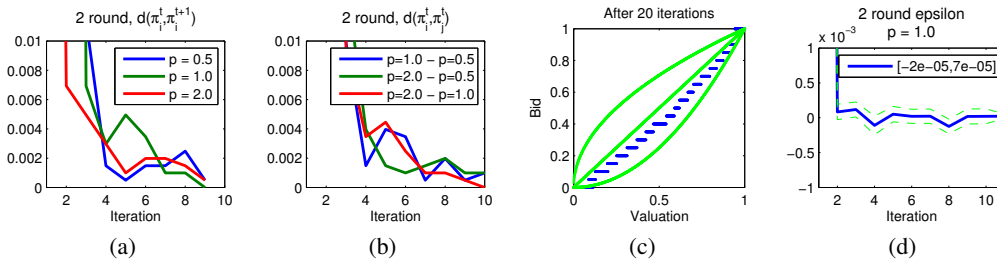

Figure 2: Convergence properties of the learning procedure in two-round MM model with 3 agents. (a),(b) evaluates convergence through $L_1$ distance of first-round bidding functions; (c) compares the learned best response (blue) with different learning procedure initializations (green). (d) plots evolution of estimated $\epsilon$-factor for learning dynamics with one specific initialization; plots for other initializations look very similar. The bracketed values in the legend give the 99% confidence bound for the $\epsilon$-factor in the final iteration, which is estimated using more sample points ($N = L = 10^9$) than previous iterations ($N = L = 10^6$).

Our iterative learning procedure is not guaranteed to converge. Nonetheless, in this experiment, our procedure not only converges with different initialization parameters $p$ (Figure 2(a)), but also converges to the same solution regardless of initial conditions (Figure 2(b)). The distance measure $d(\pi, \pi')$ between two strategies $\pi, \pi'$ in these figures is defined as the $L_1$ distance of their respective first-round bidding functions. Furthermore, the more economically meaningful measure of $\epsilon(\pi)$, measured by $\hat{\epsilon}(\pi)$, converges quickly to a negligible factor smaller than $1 \times 10^{-4}$, which is less than 0.01% of the expected bidder profit (Figure 2(d)).

All existing theoretical work on Bayesian sequential auctions with multi-unit demand is confined to two-round cases due to the increased complexity of additional rounds, but our method removes this constraint. We extend the two-round MM model into a three-round auction model,[9] and apply our learning procedure. It requires more iterations for our algorithm to converge in this set up, but it again converges to a rather stable $\epsilon$-BNE regardless of initial conditions. The final $\epsilon$-factor is smaller than 0.5% of expected bidder profit (Figure 3(d)). Although $d(\pi, \pi')$ no longer fully summarizes strategy differences, it still strongly indicates that the learning procedure converges to very similar strategies regardless of initial conditions (Figure 3(b)).

## 6 Related Work

On the theoretical side, Weber [23] derived equilibrium strategies for a basic model in which $n$ bidders compete in $k$ first or second price auctions, but bidders are assumed to have unit demand. Février [6] and Yao [25] studied a model where $n$ bidders have multi-unit demand, but there are only two auctions and a bidder's per-good valuation is the same across the two goods. Liu [13] and Paes Leme et al. [17] studied models of $n$ bidders with multi-unit demand where bidders have

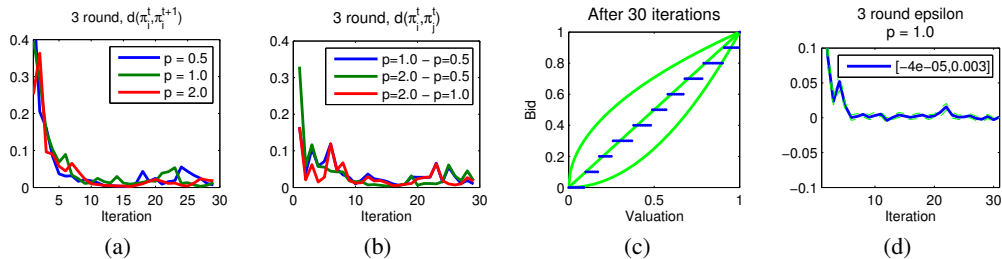

Figure 3: The same set of graphs as in Figure 2 for three round MM model with 3 agents.

complete information about opponents' valuations and perfect information about opponents' past bids. Syrgkanis and Tardos [21] extended to the case of incomplete information with unit demand.

On the computational side, Rabinovich et al. [19] generalized fictitious play to finite-action incomplete information games and applied their technique to simultaneous second-price auctions with utilities expressible as linear functions over a one-dimensional type space. Cai and Wurman [3] take a heuristic approach to finding equilibria for sequential auctions with incomplete information; opponent valuations are sampled to create complete information games, which are solved with dynamic programming and a general game solver, and then aggregated into mixed behavior strategies to form a policy for the original incomplete information game. Fatima et al. [5] find equilibrium bidding strategies in sequential auctions with incomplete information under various rules of information revelation after each round. Additional methods of computing equilibria have been developed for sequential games outside the context of auctions: Ganzfried and Sandholm [7] study the problem of computing approximate equilibria in the context of poker, and Mostafa and Lesser [15] describe an anytime algorithm for approximating equilibria in general incomplete information games.

From a decision-theoretic perspective, the bidding problem for sequential auctions was previously formulated as an MDP in related domains. In Boutilier et al. [2], an MDP is created where distinct goods are for sold consecutively, complementarities exist across goods, and the bidder is budget-constrained. A similar formulation was studied in Greenwald and Boyan [8], but without budget constraints. There, purchasing costs were models as negative rewards, significantly reducing the size of the MDP's state space. Lee et al. [12] represent multi-round games as iterated semi-net-form games, and then use reinforcement learning techniques to find $K$-level reasoning strategies for those games. Their experiments are for two-player games with perfect information about opponent actions, but their approach is not conceptually limited to such models.

## 7    Conclusion

We presented a two step procedure (predict and optimize) for finding approximate equilibria in a class of complex sequential auctions in which bidders have incomplete information about opponents' types, imperfect information about opponents' bids, and demand multiple goods. Our procedure is applicable under numerous pricing rules, allocation rules, and information-revelation policies. We evaluated our method on models with analytically derived equilibria and on an auction domain in which analytical solutions were heretofore unknown. Our method was able to both show that the known equilibrium for one model was not strict and guided our own analytical derivation of the non-strict set of equilibria. For a more complex auction with no known analytical solutions, our method converged to an approximate equilibria with an $\epsilon$-factor less than $10^{-4}$, and did so robustly with respect to initialization of the learning procedure. While we achieved fast convergence in the MM model, such convergence is not guaranteed. The fact that our procedure converged to nearly identical approximate equilibria even from different initializations is promising, and further exploring convergence properties in this domain is a direction for future work.

**Acknowledgements**    This research was supported by U.S. National Science Foundation Grants CCF-0905139 and IIS-1217761. The authors (and hence, the paper) benefited from lengthy discussions with Michael Wellman, Michael Littman, and Victor Naroditskiy. Chris Amato also provided useful insights, and James Tavares contributed to the code development.

## Footnotes

[1] See http://wireless.fcc.gov/auctions/default.htm?job=auctions_all.

[2] See http://www.floraholland.com/en/.

[3]Note that this result also generalizes to non-symmetric equilibria: we would calculate a vector of induced transition probabilities (one per bidder), given a vector of MDP assessments, (one per bidder), instead of assuming that each bidder abides by the same assessment. Similarly, stability would need to be defined in terms of a vector of MDP assessments. We present our theoretical results in terms of symmetric equilibria for notational simplicity, and because we search for symmetric equilibria in Section 5.

[4]Slightly adjusted since there is error only in the transition probabilities, not in the rewards.

[7]Weber's model can be extended to any number of rounds, but is unit, not multi-unit, demand.

[8]These analytical derivations are included in supplemental material.

[9]This model is described in supplemental material.

# References

[1] R. E. Bellman. *Dynamic Programming*. Princeton University Press, Princeton, NJ, 1957.

[2] C. Boutilier, M. Goldszmidt, and B. Sabata. Sequential auctions for the allocation of resources with complementarities. In *International Joint Conference on Artificial Intelligence*, volume 16, pages 527–534. Lawrence Erlbaum Associates LTD, 1999.

[3] G. Cai and P. R. Wurman. Monte Carlo approximation in incomplete information, sequential auction games. *Decision Support Systems*, 39(2):153–168, Apr. 2005.

[4] A. Cournot. *Recherches sur les Principes Mathematics de la Theorie la Richesse*. Hachette, 1838.

[5] S. S. Fatima, M. Wooldridge, and N. R. Jennings. Sequential Auctions in Uncertain Information Settings. *Agent-Mediated Electronic Commerce and Trading Agent Design and Analysis*, pages 16—-29, 2009.

[6] P. Février. He who must not be named. *Review of Economic Design*, 8(1):99–1, Aug. 2003.

[7] S. Ganzfried and T. Sandholm. Computing Equilibria in Multiplayer Stochastic Games of Imperfect Information. *International Joint Conference on Artificial Intelligence*, pages 140–146, 2009.

[8] A. Greenwald and J. Boyan. Bidding under uncertainty: Theory and experiments. In *Twentieth Conference on Uncertainty in Artificial Intelligence*, pages 209–216, Banff, 2004.

[9] S. M. Kakade, M. J. Kearns, and J. Langford. Exploration in metric state spaces. In *Proceedings of the 20th International Conference on Machine Learning ICML03*, 2003.

[10] B. Katzman. A Two Stage Sequential Auction with Multi-Unit Demands,. *Journal of Economic Theory*, 86(1):77–99, May 1999.

[11] P. Klemperer. *Auctions: theory and practice*. Princeton University Press, 2004.

[12] R. Lee, S. Backhaus, J. Bono, W. Dc, D. H. Wolpert, R. Bent, and B. Tracey. Modeling Humans as Reinforcement Learners : How to Predict Human Behavior in Multi-Stage Games. In *NIPS 2011*, 2011.

[13] Q. Liu. Equilibrium of a sequence of auctions when bidders demand multiple items. *Economics Letters*, 112(2):192–194, 2011.

[14] F. M. Menezes and P. K. Monteiro. Synergies and Price Trends in Sequential Auctions. *Review of Economic Design*, 8:85–98, 2003.

[15] H. Mostafa and V. Lesser. Approximately Solving Sequential Games With Incomplete Information. In *Proceedings of the AAMAS08 Workshop on Multi-Agent Sequential Decision Making in Uncertain Multi-Agent Domains*, pages 92–106, 2008.

[16] Nielsen Company. Nielsen's quarterly global adview pulse report, 2011.

[17] R. Paes Leme, V. Syrgkanis, and E. Tardos. Sequential Auctions and Externalities. In *Proceedings of the Twenty-Third Annual ACM-SIAM Symposium on Discrete Algorithms*, pages 869–886, 2012.

[18] M. Puterman. *Markov decision processes: discrete stochastic dynamic programming*. Wiley, 1994.

[19] Z. Rabinovich, V. Naroditskiy, E. H. Gerding, and N. R. Jennings. Computing pure Bayesian Nash equilibria in games with finite actions and continuous types. Technical report, University of Southampton, 2011.

[20] R. S. Sutton and A. G. Barto. *Reinforcement Learning: An Introduction*, volume 9 of *Adaptive computation and machine learning*. MIT Press, 1998.

[21] V. Syrgkanis and E. Tardos. Bayesian sequential auctions. In *Proceedings of the 13th ACM Conference on Electronic Commerce*, pages 929–944. ACM, 2012.

[22] W. Vickrey. Counterspeculation, Auctions, and Competitive Sealed Tenders. *Journal of Finance*, 16(1):8–37, 1961.

[23] R. J. Weber. Multiple-Object Auctions. In R. Engelbrecht-Wiggans, R. M. Stark, and M. Shubik, editors, *Competitive Bidding, Auctions, and Procurement*, pages 165–191. New York University Press, 1983.

[24] M. Wellman, E. Sodomka, and A. Greenwald. Self-confirming price prediction strategies for simultaneous one-shot auctions. In *The Conference on Uncertainty in Artificial Intelligence (UAI)*, 2012.

[25] Z. Yao. Sequential First-Price Auctions with Multi-Unit Demand. Technical report, Discussion paper, UCLA, 2007.

